# An Analog VLSI Model of Periodicity Extraction

**André van Schaik**
Computer Engineering Laboratory
J03, University of Sydney, NSW 2006
Sydney, Australia
*andre@ee.usyd.edu.au*

## Abstract

This paper presents an electronic system that extracts the periodicity of a sound. It uses three analogue VLSI building blocks: a silicon cochlea, two inner-hair-cell circuits and two spiking neuron chips. The silicon cochlea consists of a cascade of filters. Because of the delay between two outputs from the silicon cochlea, spike trains created at these outputs are synchronous only for a narrow range of periodicities. In contrast to traditional band-pass filters, where an increase in selectivity has to be traded off against a decrease in response time, the proposed system responds quickly, independent of selectivity.

## 1 Introduction

The human ear transduces airborne sounds into a neural signal using three stages in the inner ear's cochlea: (i) the mechanical filtering of the Basilar Membrane (BM), (ii) the transduction of membrane vibration into neurotransmitter release by the Inner Hair Cells (IHCs), and (iii) spike generation by the Spiral Ganglion Cells (SGCs), whose axons form the auditory nerve. The properties of the BM are such that close to the entrance of the cochlea (the base) the BM is most sensitive to high frequencies and at the apex the BM responds best to low frequencies. Along the BM the best-frequency decreases in an exponential manner with distance along the membrane. For frequencies below a given point's best-frequency the response drops off gradually, but for frequencies above the best-frequency the response drops off rapidly (see Fig. 1b for examples of such frequency-gain functions).

An Inner Hair Cell senses the local vibration of a section of the Basilar Membrane. The intracellular voltage of an IHC resembles a half-wave-rectified version of the local BM vibration, low-pass filtered at 1 kHz. The IHC voltage has therefore lost it's AC component almost completely for frequencies above about 4 kHz. Well below this frequency, however, the IHC voltage has a clear temporal structure, which will be reflected in the spike trains on the auditory nerve.

These spike trains are generated by the spiral ganglion cells. These SGCs spike with a probability roughly proportional to the instantaneous inner hair cell voltage. Therefore, for the lower sound frequencies, the spectrum of the input waveform is not only encoded in the form of an average spiking rate of different fibers along the

cochlea (place coding), but also in the periodicity of spiking of the individual auditory nerve fibers. It has been shown that this periodicity information is a much more robust cue than the spatial distribution of average firing rates [1]. Some periodicity information can already be detected at intensities 20 dB below the intensity needed to obtain a change in average rate. Periodicity information is retained at intensities in the range of 60-90 dB SPL, for which the average rate of the majority of the auditory nerve fibers is saturated. Moreover, the positions of the fibers responding best to a given frequency move with changing sound intensity, whereas the periodicity information remains constant. Furthermore, the frequency selectivity of a given fiber's spiking rate is drastically reduced at medium and high sound intensities. The robustness of periodicity information makes it likely that the brain actually uses this information.

## 2   Modelling periodicity extraction

Several models have been proposed that extract periodicity information using the phase encoding of fibers connected to the same inner hair cell or that use the synchronicity of firing on auditory nerve fibers connected to different inner hair cells (see [2] for 4 examples of these models). The simplest of the phase encoding schemes correlate the output of the cochlea at a given position with a delayed version of itself. It is easy to see that for pure tones, the comparison $sin(2 \pi f t) = sin(2 \pi f (t - \Delta))$ is only true for frequencies that are a multiple of $1/\Delta$, i.e., for these frequencies the signals are in perfect synchrony and thus perfectly correlated. We can adapt the delay $\Delta$ to each cochlear output, so that $1/\Delta$ equals the best frequency of that cochlear output. In this case higher multiples of $1/\Delta$ will be suppressed due to the very steep cut-off of the cochlear filters for frequencies above the best frequency. Each synchronicity detector will then only be sensitive to the best frequency of the filter to which it is connected. If we code the direct signal and the delayed signal with two spike trains, with one spike per period at a fixed phase each, it becomes a very simple operation to detect the synchronicity. A simple digital AND operator will be enough to detect overlap between two spikes. These spikes will overlap perfectly when $f = 1/\Delta$, but some overlap will still be present for frequencies close to $1/\Delta$, since the spikes have a finite width. The bandwidth of the AND output can thus be controlled by the spike width.

It is possible to create a silicon implementation of this scheme using an artificial cochlea, an IHC circuit, and a spiking neuron circuit together with additional circuits to create the delays. A chip along these lines has been developed by John Lazzaro [3] and functioned correctly. A disadvantage of this scheme, however, is the fact that the delay associated with a cochlear output has to be matched to the inverse of the best frequency of that cochlear output. For a cochlea whose best frequency changes exponentially with filter number in the cascade from 4 kHz (the upper range of phase locking on the auditory nerve) to 100 Hz, we will have to create delays that range from 0.25 ms to 10 ms. In the brain, such a large variation in delays is unlikely to be provided by an axonal delay circuit because it would require an excessively large variation in axon length.

A possible solution comes from the observation that the phase of a pure tone of a given frequency on the basilar membrane increases from base to apex, and the phase changes rapidly around the best frequency. The silicon cochlea, which is implemented with a cascade of second-order low-pass filters (Fig. 1a), also functions as a delay line, and each filter adds a delay which corresponds to $\pi/2$ at the cut-off frequency of that filter. If we assume that filter i and filter i-4 have the same cut-off frequency (which is not the case), the delay between the output of both filters will correspond to a full period ($2\pi$) at the cut-off frequency.

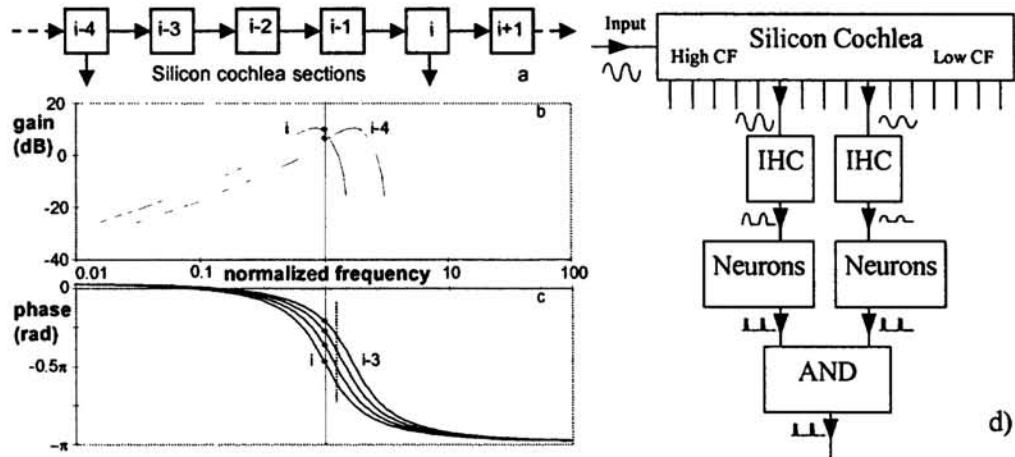

Figure 1: a) Part of a silicon cochlea. Each section contains a second-order low-pass filter and a derivator; b) accumulated gain at output i and i-4; c) phase curves of the individual stages between output i and output i-4; d) proposed implementation of the periodicity extraction model.

In reality, the filters along the cochlea will have different cut-off frequencies, as shown in Fig. 1. Here we show the *accumulated* gain at the outputs i and i-4 (Fig. 1b), and the delay added by each *individual* filter between these two outputs (Fig. 1c) as a function of frequency (normalized to the cut-off frequency of filter i). The solid vertical line represents this cut-off frequency, and we can see that only filter i adds a delay of $\pi/2$, and the other filters add less. However, if we move the vertical line to the right (indicated by the dotted vertical line), the delay added by each filter will increase relatively quickly, and at some frequency slightly higher than the cut-off frequency of filter i, the sum of the delays will become $2\pi$ (dashed line). At this frequency neither filter i nor filter i-4 has maximum gain, but if the cut-off frequency of both filters is not too different, the gain will still be high enough for both filters at the correlator frequency to yield output signals with reasonable amplitudes.

The improved model can be implemented using building blocks as shown in Fig. 1d. Each of these building blocks have previously been presented (refer to [4] for additional details). The silicon cochlea is used to filter and delay the signal, and has been adjusted so that the cut-off frequency decreases by one octave every twenty stages, so that the cut-off frequencies of neighboring filters are almost equal. The IHC circuit half-wave rectifies the signal in the implementation of Figure 1d. The low-pass filtering of the biological Inner Hair Cell can be ignored for frequencies below the approximately 1kHz cut-off frequency of the cell. Since we limited our measurements to this range, the low-pass filtering has not been modeled by the circuit. Two chips containing electronic leaky-integrate-and-fire neurons have been used to create the two spike trains. In the first series of measurements, each chip generates exactly one spike per period of the input signal. A final test will set the 32 neurons on each chip to behave more like biological spiral ganglion cells and the effect on periodicity extraction will be shown. A digital AND gate is used to compare the output spikes of the two chips, and the spike rate at the output of the AND gate is the measure of activity used.

## 3   Test results

The first experiment measures the number of spikes per second at the output of the AND gate as a function of input frequency, using different cochlear filter combinations. Twelve filter pairs have been measured, each combining a filter

output with the output of a filter four sections earlier in the cascade. The best frequency of the filter with the lowest best frequency of the pairs ranged from 200 Hz to 880 Hz. The results are shown in Fig. 2a.

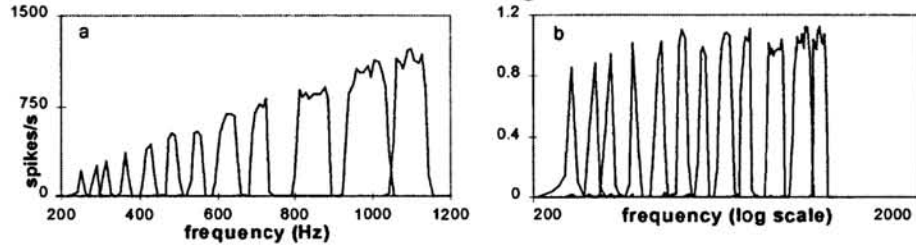

Figure 2: a) measured output rate at different cochlear positions, and b) spike rate normalized to best input frequency, plotted on a log frequency scale.

The maximum spike rate increases approximately linearly with frequency; this is to be expected, since we will have approximately one spike per signal period. Furthermore the best response frequencies of the filters sensitive to higher frequencies are further apart, due to the exponential scaling of the frequencies along the cochlea. Finally, a given time delay corresponds to a larger phase delay for the higher frequencies, so that the absolute bandwidth of the coincidence detectors, i.e., the range of input frequencies to which they respond, is larger. When we normalize the spike rate and plot the curves on a logarithmic frequency scale, as in Fig. 2b, we see that the best frequencies of the correlators follow the exponential scaling of the best frequencies of the cochlear filters, and that the relative bandwidth is fairly constant.

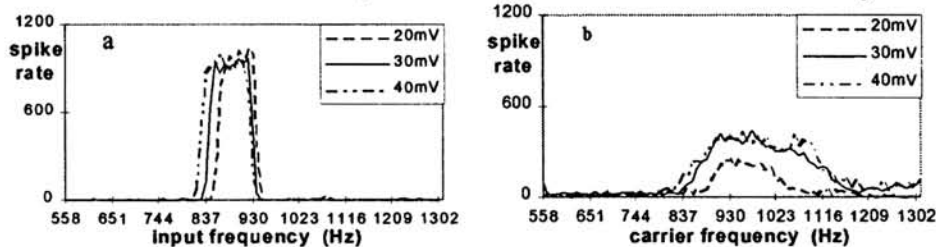

Figure 3: Frequency selectivity for different input intensities. a) pure tones b) AM signals.

Using the same settings as in the previous experiment, the output spike rate of the system for different input amplitudes has been measured, using the cochlear filter pair with best frequencies of 710 Hz and 810 Hz. In principle, the amplitude of the input signal should have no effect on the output of the system, since the system only uses phase information. However, this is only true if the spikes are always created at the same phase of the output signal of the cochlear filters, for instance at the peak, or the zero crossing. Fig. 3 shows however that the resulting filter selectivity shifts to lower frequencies for higher intensity input signals.

This is a result of the way the spikes are created on the neuron chip. The neurons have been adjusted to spike once per period, but the phase at which they spike with respect to the half-wave-rectified waveform depends on the integration time of the neuron, which is the time needed with a given input current to reach the spike threshold voltage from the zero resting voltage. This time depends on the amplitude of the input current, which in turn is proportional to the amplitude of the input signal. Since the amplitude gain of the two cochlear filters used is not the same, the amplitude of the current input to the two neuron chips is different. Therefore, they do not spike at the same phase with respect to their respective input waveforms. This causes the frequency selectivity of the system to shift to lower frequencies with increasing intensity. However, this is an artifact of the spike generation used to

simplify the system. On the auditory nerve, spikes arrive with a probability roughly proportional to the half-wave rectified waveform. The most probable phase for a spike is therefore always at the maximum of the waveform, independent of intensity. In such a system, the frequency selectivity will therefore be independent of amplitude. A second advantage of coding (at least half of) the waveform in spike probability is that it does not assume that the input waveform is sinusoidal. Coding a waveform with just one spike per period can only code the frequency and phase of the waveform, but not its shape. A square wave and a sine wave would both yield the same spike train. We will discuss the "auditory-nerve-like" coding at the end of this section.

To test the model with a more complex waveform, a 930 Hz sine wave 100% amplitude-modulated at 200 Hz generated on a computer has been used. The carrier frequency was varied by playing the whole waveform a certain percentage slower or faster. Therefore the actual modulation frequency changes with the same factor as the carrier frequency. The results of this test are shown in Fig. 3b for three different input amplitudes. Compared to the measurements in Fig. 3a, we see that the filter is less selective and centered at a higher input frequency. The shift towards a higher frequency can be explained by the fact that the average amplitude of a half-wave rectified amplitude modulated signal is lower than in for a half-wave rectified pure tone with the same maximum amplitude. Furthermore, the amplitude of the positive half-cycle of the output of the IHC circuit changes from cycle to cycle because of the amplitude modulation. We have seen that the amplitude of the input signal changes the frequency for which the two spike trains are synchronous, which means that the frequency which yields the best response changes from cycle to cycle with a periodicity equal to the modulation frequency. This introduces a sort of "roaming" of the frequencies in the input signal, effectively reducing the selectivity of the filters. Finally, because of the 100% depth of the amplitude modulation, the amplitude of the input will be too low during some cycles to create a spike, which therefore reduces the total number of spikes which can coincide.

Fig. 3b shows that this model detects periodicity and not spectral content. The spectrum of a 930 Hz pure tone 100% amplitude modulated at 200 Hz contains, apart from a 930 Hz carrier component, components both at 730 Hz and 1130 Hz, with half the amplitude of the carrier component. When the speed of the waveform playback is varied so that the carrier frequency is either 765 Hz or 1185 Hz, one of these spectral side bands will be at 930 Hz, but the system does not respond at these carrier frequencies. This is explained by the fact that the periodicity of the zero crossings, and thus of the positive half cycles of the IHC output, is always equal to the carrier frequency.

Traditional band-pass filters with a very high quality factor (Q) can also yield a narrow pass-band, but their step response takes about 1.5Q cycles at the center frequency to reach steady state. The periodicity selectivity of the synchronicity detector shown in Fig. 3a corresponds to a quality factor of 14; a traditional band-pass filter would take about 21 cycles of the 930Hz input signal to reach 95% of it's final output value. Fig. 4 shows the temporal aspect of the synchronicity detection in our system. The top trace in this figure shows the output of the cochlear filter with the highest best frequency (index i-4 in Fig. 1) and the spikes generated based on this output. The second trace shows the same for the output of the cochlear filter with the lower best frequency (index i in Fig. 1). The third trace shows the output of the AND gate with the above inputs, which are slightly above its best periodicity. Coincidences are detected at the onset of the tone, even when it is not of the correct periodicity, but only for the first one or two cycles. The bottom trace shows the output of the AND gate for an input at best frequency. The system thus responds to the presence of a pure tone of the correct periodicity after only a few cycles, independent of the filters selectivity.

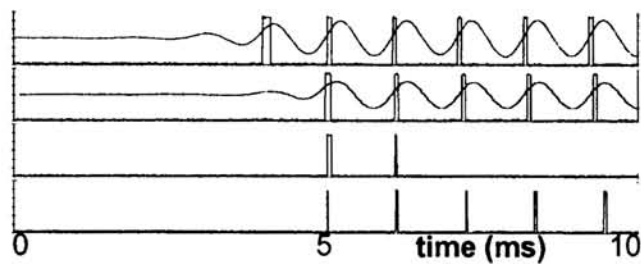

Figure 4: Oscilloscope traces of the temporal aspect of synchronicity detection. The vertical scale is 20mV per square for the cochlear outputs, the spikes are 5V in amplitude.

To show this more dramatically, we have reduced the spike width to 10 μs, to obtain a high periodicity selectivity as shown in Fig. 5a. The bandwidth of this filter is only 20 Hz at 930 Hz, equivalent to a quality factor of 46.5. A traditional filter with such a quality factor would only settle 70 cycles after the onset of the signal, whereas the periodicity detector still settles after the first few cycles, as shown in Fig. 5b. We can compare this result with the response of a classic RLC band-pass filter with a 930 Hz center frequency and a quality factor of 46.5 as shown in Fig. 6. After 18 cycles of the input signal, the output of the band-pass filter has only reached 65% of its final value. Thresholding the RLC output could signal the presence of a periodicity faster, but it would then still respond very slowly to the offset of the tone as the RLC filter will continue ringing after the offset.

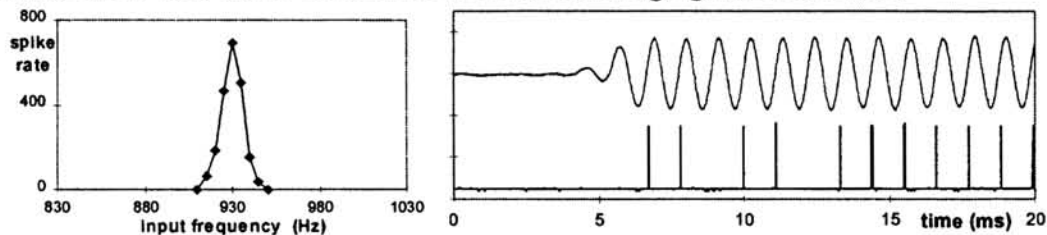

Figure 5: a) Frequency selectivity with a 10μs spike width. b) Cochlear output (top, 40 mV scale) and coincidences (bottom) for a signal at best frequency.

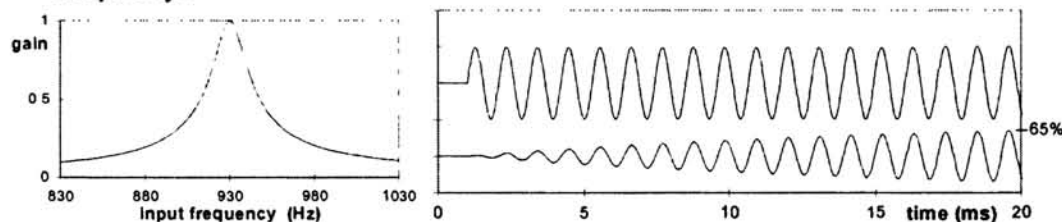

Figure 6: Simulated response of the RLC band-pass filter. a) frequency selectivity, b) transient response (scale units are 40 mV).

In the previous experiments we simplified the model to use one spike per period in order to understand the principle behind the periodicity detection. However, we have seen that this implementation leads to a shift in best periodicity with changing amplitude, because the phase at which the 'single neuron' spikes changes with intensity. Now, we will change the settings to be more realistic, so that each of the 32 neurons cannot spike at every period, and we will reduce the output gain of the IHC circuit so that the neurons receive less signal current, and thus have a lower input SNR. The resulting spike distribution is a better simulation of the spike distribution on the auditory nerve. This is shown in Fig. 7 for a group of 32 neurons stimulated by and IHC circuit connected to a single cochlear output. The bottom trace shows the sum of spikes over the 32 neurons on an arbitrary scale. When we

use this spike distribution and repeat the pure-tone detection experiment of Fig. 3a at different input intensities, we obtain the curve of Fig. 7b. Indeed, in this case, the best periodicity does not change; the curves are remarkably independent of input intensity. However, the selectivity curve is about twice as wide at the base as the ones in Fig. 3a, but the slopes of the selectivity curve rise and fall much more gradually. This means that we can easily increase the selectivity of these curves by setting a higher threshold, e.g., discarding spike rates below 70 spikes per second. Because of the steep slopes in Fig. 3a such an operation would hardly increase the selectivity for that case.

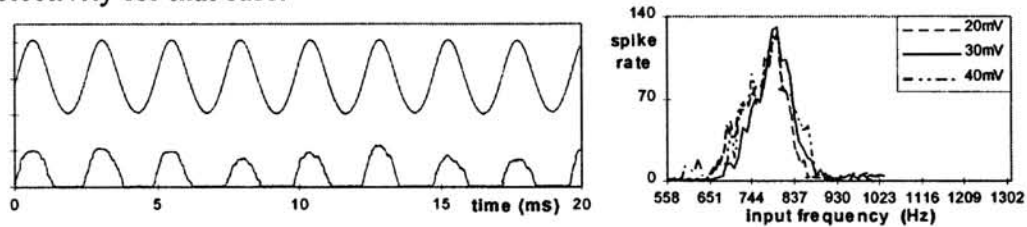

Figure 7: a) Cochlear output (top) and population average of the auditory nerve spikes (bottom); b) periodicity selectivity with auditory nerve like spike distribution.

## 4 Conclusions

In this paper we have presented a neural system for periodicity detection implemented with three analogue VLSI building blocks. The system uses the delay between the outputs at two points along the cochlea and synchronicity of the spike trains created from these cochlear outputs to detect the periodicity of the input signal. An especially useful property of the cochlea is that the delay between two points a fixed distance apart corresponds to a full period at a frequency that scales in the same way as the best frequency along the cochlea, i.e., decreases exponentially.

If we always create spikes at the same phase of the output signal at each filter, or simply have the highest spiking probability for the maximum instantaneous amplitude of the output signal, then both outputs will only have synchronous spikes for a certain periodicity, and we can easily detect this synchronicity with coincidence detectors. This system offers a way to obtain very selective filters using spikes. Even though they react to a very narrow range of periodicities, these filters are able to react after only a few periods. Furthermore, the range of periodicities it responds to can be made independent of input intensity, which is not the case with the cochlear output itself. This clearly demonstrates the advantages of using spikes in the detection of periodicity.

### Acknowledgements

The author thanks Eric Fragnière, Eric Vittoz and the Swiss NSF for their support.

### References

[1] Evans, "Functional anatomy of the auditory system," in Barlow and Mollon (editors), *The Senses*, Cambridge University Press, pp. 251-306, 1982.

[2] Seneff, Shamma, Deng, & Ghitza, *Journal of Phonetics*, Vol. 16, pp. 55-123, 1988.

[3] Lazzaro, "A silicon model of an auditory neural representation of spectral shape." *IEEE Journal of Solid-State Circuits*, Vol. 26, No. 5, pp. 772-777, 1991.

[4] van Schaik, "An Analogue VLSI Model of Periodicity Extraction in the Human Auditory System," to appear in *Analog Integrated Circuits and Signal Processing*, Kluwer, 2000.
